# Identifying Distributed Object Representations in Human Extrastriate Visual Cortex

**Rory Sayres**

Department of Neuroscience
Stanford University
Stanford, CA 94305
*sayres@stanford.edu*

**David Ress**

Department of Neuroscience
Brown University
Providence, RI 02912
*ress@brown.edu*

**Kalanit Grill-Spector**

Departments of Neuroscience and Psychology
Stanford University
Stanford, CA 94305
*kalanit@psych.stanford.edu*

## Abstract

The category of visual stimuli has been reliably decoded from patterns of neural activity in extrastriate visual cortex [1]. It has yet to be seen whether object identity can be inferred from this activity. We present fMRI data measuring responses in human extrastriate cortex to a set of 12 distinct object images. We use a simple winner-take-all classifier, using half the data from each recording session as a training set, to evaluate encoding of object identity across fMRI voxels. Since this approach is sensitive to the inclusion of noisy voxels, we describe two methods for identifying subsets of voxels in the data which optimally distinguish object identity. One method characterizes the reliability of each voxel within subsets of the data, while another estimates the mutual information of each voxel with the stimulus set. We find that both metrics can identify subsets of the data which reliably encode object identity, even when noisy measurements are artificially added to the data. The mutual information metric is less efficient at this task, likely due to constraints in fMRI data.

## 1 Introduction

Humans and other primates can perform fast and efficient object recognition. This ability is mediated within a large extent of occipital and temporal cortex, sometimes referred to as the ventral processing stream [10]. This cortex has been examined using electrophysiological recordings, optical imaging techniques, and a variety of neuroimaging techniques including functional magnetic resonance imaging (fMRI) [refs]. With fMRI, these regions can be reliably identified by their strong preferential response to intact objects over other visual stimuli [9,10].

The functional organization of object-selective cortex is unclear. A number of regions have been identified within this cortex, which preferentially respond to particular categories of images [refs]; it has been proposed that these regions are specialized for processing visual information about those categories [refs]. A recent study by Haxby and

colleagues [1] found that the category identity of different stimuli could be decoded from fMRI response patterns, using a simple classifier in which half of each data set was used as a training set and half as a test set. These results were interpreted as evidence for a distributed representation of objects across ventral cortex, in which both positive and negative responses contribute information about object identity. It is not clear, however, to what extent information about objects is processed at the category level, and to what extent it reflects individual object identity, or features within objects [1,8].

The study in [1] is one of a growing number of recent attempts to decode stimulus identity by examining fMRI response patterns across cortex [1-4]. fMRI data has particular advantages and disadvantages for this approach. Among its advantages are the ability to make many measurements across a large extent of cortex in awake, behaving humans. Its disadvantages include temporal and spatial resolution constraints, which limit the number of trials that may be collected; the ability to examine trial-by-trial variation; and potentially limit the localization of small neuronal populations. A further potential disadvantage arises from the little-understood functional organization of object-selective cortical regions. Because it is not clear which parts of this cortex are involved in representing different objects and which aren't, analyses may include fMRI image locations (voxels) which are not involved in object representation.

The present study addresses a number of these questions by examining the response patterns across object-selective cortex to a set of 12 individual object images, using high-resolution fMRI. We sought to address the following experimental questions: (1) Can individual object identity be decoded from fMRI responses in object-selective cortex? (2) How can one identify those subsets of fMRI voxels which reliably encode identity about a stimulus, among a large set of potentially unrelated voxels? We adopt a similar approach to that described in [1], subdividing each data set into training and test subsets, and evaluate the efficiency of a set of voxels in discriminating object identity among the 12 possible images with a simple winner-take-all classifier. We then describe two metrics from which to identify sets of voxels which reliably discriminate different objects. The first metric estimates the replicability of voxels to each stimulus between the training and the test data. The second metric estimates the mutual information each voxel has with the stimulus set.

## 2 Experimental design and data collection

Our experimental design is summarized in Figure 1. We chose a stimulus set of 12 line drawings of different object stimuli, shown in Figure 1a. These objects can be readily categorized as faces, animals, or vehicles; these categories have been previously identified as producing distinct patterns of blood-oxygenation-level-dependent (BOLD) response in object-selective cortex [10]. This allows us to compare category and object identity as potential explanatory factors for BOLD response patterns. Further, the use of black-and-white line drawings reduces the number of stimulus features which differentiate the stimuli, such as spatial frequency bands.

A typical trial is illustrated in Figure 1b. We presented one of the 12 object images to the subject within the foveal 5 degrees of visual field for 2 sec, then masked the image with a scrambled version of a random image for 10 sec. These scrambled images are known to produce minimal response in our regions of interest [11], and serve as a baseline condition for these experiments. Each scan contained one trial per image, presented in a randomized order. We ran 10-15 event-related scans for each scanning session. This allowed us to collect full hemodynamic responses to each image, which in BOLD signal lags several seconds after stimulus onset. In this way we were able to analyze trial-by-trial variations in response to different images, without the analytic and design restrictions involved in analyzing fMRI data with more closely-spaced trials [5]. This feature was essential for computing the mutual information of a voxel with the stimulus set.

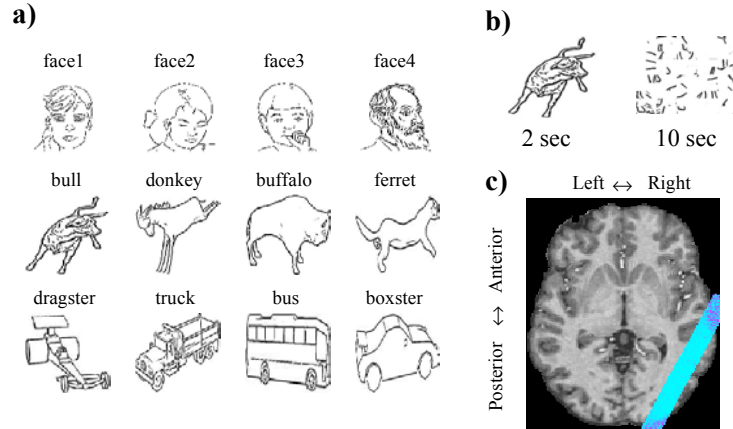

Figure 1: Experimental Design. **(a)** The 12 object stimuli used. **(b)** Example of a typical trial. **(c)** Depiction of imaged region during one session. The image is an axial slice from a T1-weighted anatomical image for one subject. The blue region shows the region imaged at high resolution. The white outlines show gray matter within the imaged area.

We obtained high-resolution fMRI images at 3 Tesla using a spiral-out protocol. We used a custom-built receive-only surface coil. This coil was small and flexible, with a 7.5 cm diameter, and could be placed on a subject's skull directly over the region to be imaged. Because of the restricted field of view of this coil, we imaged only right hemisphere cortex for these experiments. We imaged 4 subjects (1 female), each of whom participated in multiple recording sessions. For each recording session, we imaged 12 oblique slices, with voxel dimensions of 1 x 1 x 1 mm and a frame period of 2 seconds. (More typical fMRI resolutions are around 3 x 3 x 3 mm–3x3x6 mm, at least 27 times lower in resolution.) A typical imaging prescription, superimposed over a high-resolution T1-weighted anatomical image, is shown in Figure 1c.

Functional data from these experiments are illustrated in Figure 2. Within each session, we identified object-selective voxels by applying a general linear model to the time series data, estimating the amplitude of BOLD response to different images [5]. We then computed contrast maps representing T tests of response of different images against the baseline scrambled condition. An example of voxels localized in this way is illustrated in Figure 2a, superimposed over mean T1-weighted anatomical images for two slices. Our criterion for defining object-selective voxels was that a voxel needed to respond to at least one of the 12 stimulus images relative to baseline with a significance level of $p \leq 0.001$. Each data set contained between 600 and 2500 object-selective voxels.

The design of our surface coil, combined with its proximity to the imaged cortex, allowed us to observe significant event-related responses within single voxels. Figure 2b shows peri-stimulus time courses to each image from four sample voxels. These responses are summarized by subtracting the mean BOLD response after stimulus onset with the response during the baseline period, as illustrated in Figure 2c. In this way we can summarize a data set as a matrix $\mathbf{A}$ of response amplitudes to different voxels, where $\mathbf{A}_{i,j}$ represents the response to the $i$th image of the $j$th voxel. These responses are statistically significant (T test, $p < 0.001$) for many stimuli, yet the voxels are heterogeneous in their responses—different voxels respond to different stimuli. This response diversity prompts the questions of deciding which sets of responses, if any, are informative of image identity.

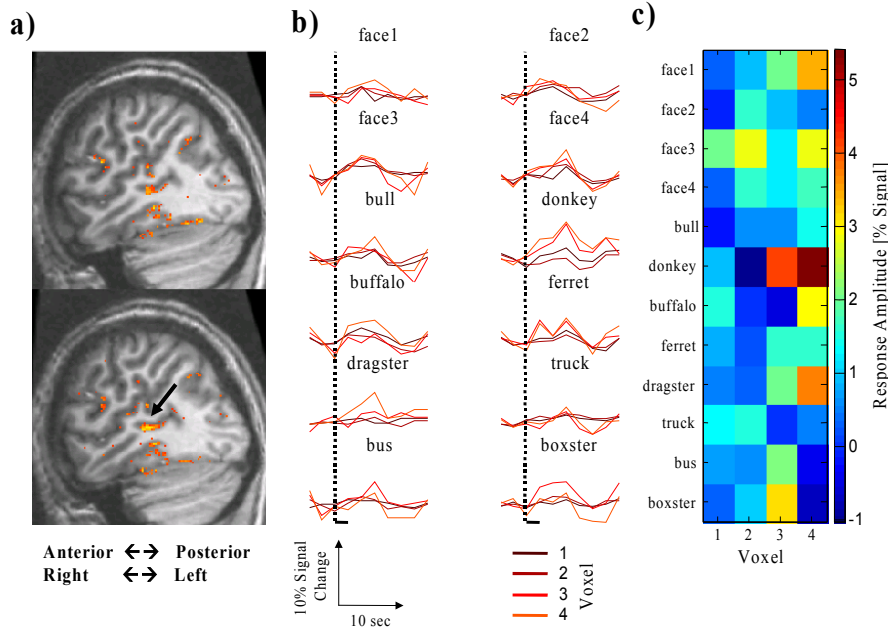

Figure 2: Experimental Data. **(a)** T1-weighted anatomical images from a sample session, with object-selective voxels indicated in orange. **(b)** Mean peristimulus time courses from 4 object-selective voxels in the lower slice of **(a)** (locations indicated by arrow), for each image. Dotted lines indicate trial onset; dark bars at bottom indicate stimulus presentation duration. Scale bars indicate 10 seconds duration and 10 percent BOLD signal change relative to baseline. **(c)** Mean response amplitudes from the voxels depicted in **(b)**, represented as a set of column vectors for each voxel. Color indicates mean amplitude during post-stimulus period relative to pre-stimulus period.

## 3    Winner-take-all classifier

Given a set of response amplitudes across object-selective voxels, how can we characterize the discriminabilty of responses to different stimuli? This question can be answered by constructing a classifier, which takes a set of responses to an unknown stimulus, and compares it to a training set of responses to known stimuli. This general approach has been successfully applied to fMRI responses in early visual cortex [3-4], object-selective cortex [1], and across multiple cortical regions [2].

For our classifier, we adopt the approach used in [1], with a few refinements. As in the previous study, we subdivide each data set into a training set and a test set, with the training set representing odd-numbered runs and the test set representing even-numbered runs. (Since each run contains one trial per image, this is equivalent to using odd- and even-numbered trials). We construct a training matrix, $A_{training}$, in which each row represents the response across voxels to a different image in the training data set. We construct a second matrix, $A_{test}$, which contains the responses to different images during the test set. These matrices are illustrated for one data set in Figure 3a. Each row of $A_{test}$ is considered to be the response to an unknown stimulus, and is compared to each of the rows in $A_{training}$. The overall performance of the classifier is evaluated by its success rate at classifying test responses based on the correlation to training responses.

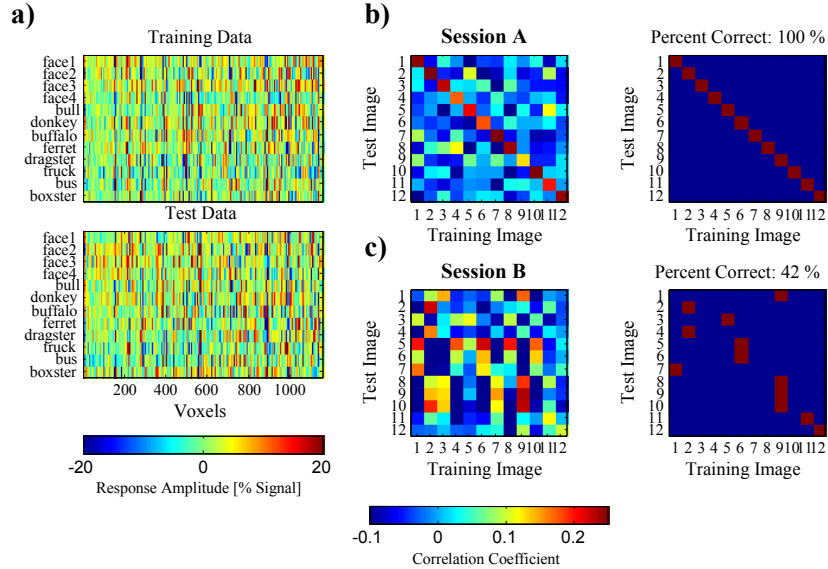

Figure 3: Illustration of winner-take-all classifier for two sample sessions. **(a)** Response amplitudes for all object-selective voxels for the training *(top)* and test *(bottom)* data sets, for one recording session. **(b)** Classifier results for the same session as in **(a)**. *Left:* Correlation matrix between the training and test sets. *Right:* Results of the winner-take-all algorithm. The red square in each row represents the image from the test set that produced the highest correlation with the training set, and is the "guess" of the classifier. The percent correct is evaluated as the number of guesses that lie along the diagonal (the same image in the training and test sets produces the highest correlation). **(c)** Results for a second session, in the same format as **(b)**.

We evaluate classifier performance with a winner-take-all criterion, which is more conservative than the criterion in [1]. First, a correlation matrix $\mathbf{R}$ is constructed containing correlation coefficients for each pairwise comparison of rows in $\mathbf{A}_{\text{training}}$ and $\mathbf{A}_{\text{test}}$ (shown on the left in Figure 3b and 3c for two data sets). The element $\mathbf{R}_{\text{i,j}}$ represents the correlation coefficient between row $i$ of $\mathbf{A}_{\text{test}}$ and row $j$ of $\mathbf{A}_{\text{training}}$. Then, for each row in the correlation matrix, the classifier "guesses" the identity of the test stimulus by selecting the element with the highest coefficient (shown on the right in Figure 3b and 3c). Correct guesses lie along the diagonal of this matrix, $\mathbf{R}_{\text{i,i}}$.

The previously-used method evaluated classifier performance by successively pairing off the correct stimulus with incorrect stimuli from the training set [1]. With this criterion, responses from the test set which do not correlate maximally with the same stimulus in the training set might still lead to high classifier performance. For instance, if an element $\mathbf{R}_{\text{i,i}}$ is larger than all but one coefficient in row $i$, pairwise comparisons would reveal correct guesses for 10 out of 11 comparisons, or 91% correct, while the winner-take-all criterion would consider this 0%. This conservative criterion reduces chance performance from 1/2 to 1/12, and ensures that high classifier performance reflects a high level of discriminability between different stimuli, providing a stringent test for decoding.

## 4 Identifying voxels which distinguish objects

When we examined response patterns across all object-selective voxels, we observed high levels of classifier performance from some recording sessions, as shown in Session A in Figure 3. Many sessions, however, were more similar to Session B: limited success at decoding object identity when using all voxels.

For both cases, a relevant question is the extent to which information is contained within a subset of the selected voxel. The distributed representation implied in Session A may be driven by only a few informative voxels; conversely, excessively noisy or unrelated activity from other voxels may be affected classifier performance on Session B. This is of particular concern given that the functional organization of this cortex is not well understood. In addition to using such classifiers to test a hypothesis that a pre-defined region of interest can discriminate stimuli, it would be highly useful to use the classifier to identify cortical regions which represent a stimulus.

To identify subsets of the data which reliably represent different stimuli, we search among the set of object-selective voxels using two metrics to rank voxels: (1) The reliability of each voxel between the training and test data subsets; and (2) The mutual information of each voxel with the stimulus set.

## 4.1    Voxel reliability metric

The voxel reliability metric is computed for each voxel by taking the vectors of 12 response amplitudes to each stimulus in the training and test sets, and calculating their correlation coefficient. Voxels with high reliability will have high values for the diagonal elements in the **R** correlation matrix, but this does not place constraints on correlations for the off-diagonal comparisons. For instance, persistently active and nonspecific voxels (such as might be expected from draining veins or sinuses) would have high voxel reliability, but also high correlation for all pairwise comparisons between stimuli in test and training sets, so as not to guarantee high classifier performance.

## 4.2    Mutual information metric

The mutual information for a voxel is computed as the difference between the overall entropy of the voxel and the "noise entropy", the sum over all stimuli of the entropy of the voxel given each stimulus [6]:

$$I_m = H - H_{noise} = -\sum_r P(r)\log_2 P(r) + \sum_{s,r} P(s)P(r|s)\log_2 P(r|s) \quad (1)$$

In this formula, P(r) represents the probability of observing a response level r and P(r|s) represents the probability of observing response r given stimulus s. Computing these probabilities presents a difficulty for fMRI data, since an accurate estimate requires many trials. Given the hemodynamic lag of 9-16 sec inherent to measuring BOLD signal, and the limitations of keeping a human observer in an MRI scanner before motion artifacts or attentional drifts confound the signals, it is difficult to obtain many trials over which to evaluate different response probabilities. There are two possible solutions to this: find ways of obtaining large number of trials, e. g. through co-registering data across many sessions; and reduce the number of possible response bins for the data. While the first option is an area of active pursuit for us, we will focus here on the second approach.

Given the low number of trials per image, we reduce the number of possible response levels to only two bins, 0 and 1. This allows for a wider range of possible values for P(r) and P(r|s) at the expense of ignoring potential information contained in varying response levels. Given these two bins, the next question is deciding how to threshold responses to decide if a given voxel responded significantly (r=1) or not (r=0) on a given trial. Since we do not have an *a priori* hypothesis about the value of this threshold, we choose it separately for each voxel, such that it maximizes the mutual information of that voxel. This approach has been used previously to reduce free parameters while developing artificial recognition models[7].

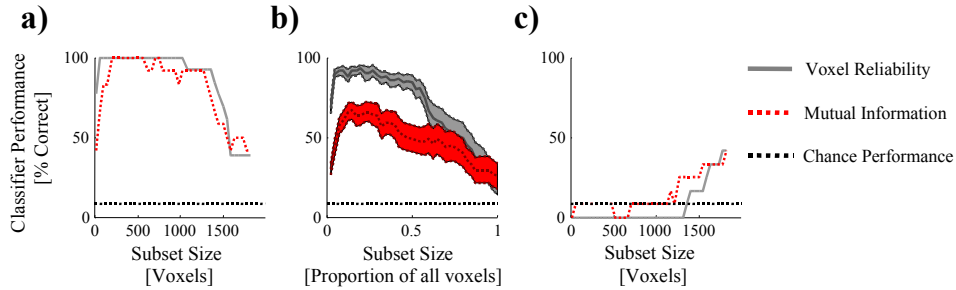

Figure 4: Comparison of metrics for identifying reliable subsets of voxels in data sets. **(a)** Performance on winner-take-all classifier of different-sized subsets of one data set ("Session B" in Figure 3), sorted by voxel reliability *(gray, solid)* and mutual information *(red, dashed)* metrics. **(b)** Performance of the two metrics across 12 data sets. Each curve represents the mean (thick line) ± standard error of the mean across data sets. **(c)** Performance on data set from **(a)** when reverse-sorting voxels by each metric. Dotted black line indicates chance performance.

After ranking each voxel with the two metrics, we evaluated how well these voxels found reliable object representations. To do this, we sorted the voxels in descending order according to each metric; selected progressively larger subsets of voxels, starting with the 10 highest-ranked voxels and proceeding to the full set of voxels; and evaluated performance on the classifier for each subset. Results of these analyses are summarized in Figure 4. Figure 4a shows performance curves for the two sortings on data from the "Session B" data set illustrated in Figure 3. As can be seen, while performance using all voxels is at 42% correct, by removing voxels, performance quickly reaches 100% using the reliability criterion. The mutual information metric also converges to 100%, albeit slightly more slowly. Also note that for very small subset sizes, performance decreases again: correct discrimination requires information distributed across a set of voxels.

Finally, we repeated our analyses across 12 data sets collected from 4 subjects. Figure 4c shows the mean performance across sessions for the two metrics. These curves are normalized by the proportion of total available voxels for each data set. Overall, the voxel reliability metric was significantly better at identifying subsets of voxels which could discriminate object identity, although both metrics performed significantly better than the 1/12 chance performance at the classifier task, and both produced pronounced improvements in performance for smaller subsets compared to using the entire data sets. Note that simply removing voxels does not guarantee the better performance on the classifier. If the voxels are sorted in reverse order, starting with e. g. the lowest values of voxel reliability or mutual information, subsets containing half the voxels are consistently at or below chance performance (Figure 4c).

## 5   Summary and conclusions

Developing and training classifiers to identify cognitive states based on fMRI data is a growing and promising approach for neuroscience [1-4]. One drawback to these methods, however, is that they often require prior knowledge of which voxels are involved in specifying a cognitive state, and which aren't. Given the poorly-understood functional organization of the majority of cortex, an important goal is to develop methods to search across cortex for regions which represent such states. The results described here represent one step in this direction.

Our voxel-ranking metrics successfully identified subsets of object-selective voxels

which discriminate object identity. This demonstrates the feasibility of adapting classifier methods to search across cortical regions. However, these methods can be refined considerably. The most important improvement is providing a larger set of trials from which to compute response probabilities. This is currently being pursued by combining data sets from multiple recording sessions in a reference volume. Given more extensive data, the set of possible response bins can be increased from the current binary set, which should improve performance of our mutual information metric.

Our results also have several implications for object recognition. We found a high ability to discriminate between individual images in our data sets. Moreover, this discrimination could be performed with sets of voxels of widely varying sizes. For some sessions, perfect discrimination could be achieved using all object-selective voxels, which number in the thousands (Figure 3a, 3b); for many others, perfect discrimination was possible using subsets as small as a few dozen voxels. This has implications for the distributed nature of object representation in extrastriate cortex. However, it raises the question of identifying redundant information within these representations. The distributed representations may reflect functionally distinct areas which are processing different aspects of each stimulus, as in earlier visual cortex. Mutual information approaches have succeeded at identifying redundant coding of information in other sensory areas [10], and can be tested on the known functional subdivisions in early visual cortex. In this way, we can use intuitions generated by ideal observers of the data, such as the classifier described here, and apply them to understanding how the brain processes this information.

**Acknowledgments**

We would like to thank Gal Chechik and Brian Wandell for input on analysis techniques. This work was supported by NEI National Research Service Award 5F31EY015937-02 to RAS, and a research grant 2005-05-111-RES from the Whitehall Foundation to KGS.

**References**

[1] Haxby JV, Gobbini MI, Furey ML, Ishai A, Schouten JL, and Pietrini P. (2001) Distributed and overlapping representations of faces and objects in ventral temporal cortex. *Science* 293:2425-30.

[2] Wang X, Hutchinson R, and Mitchell TM (2004) Training fMRI classifiers to distinguish cognitive states across multiple subjects. In S. Thrun, L. Saul and B. Scholköpf (eds.), Advances in Neural Information Processing Systems 16. Cambridge, MA: MIT Press.

[3] Kamitani Y and Tong F. (2005) Decoding the visual and subjective contents of the human brain. *Nat Neurosci.* 8:679-85.

[4] Haynes JD and Rees G. (2005) Predicting the orientation of invisible stimuli from activity in human primary visual cortex. *Nat Neurosci.* 8:686-691.

[5] Burock MA and Dale AM. (2000) Estimation and Detection of Event-Related fMRI Signals with temporally correlated noise: a statistically efficient and unbiased approach. *Human Brain Mapping* 11:249-260.

[6] Abbott L and Dayan P (2001) Theoretical Neuroscience. Cambridge, MA: MIT Press.

[7] Ullman S, Vidal-Naquet M, and Sali E. Visual features of intermediate complexity and their use in classification. *Nat Neurosci*. 5(7):682-7.

[8] Tsunoda K, Yamane Y, Nishizaki M, and Tanifuji M. (2001) Complex objects are represented in macaque inferotemporal cortex by the combination of feature columns. *Nat Neurosci.* 4:832-8.

[9] Grill-Spector K, Kushnir T, Hendler T, and Malach R. (2000) The dynamics of object-selective activation correlate with recognition performance in humans. *Nat Neurosci.* 3:837-43.

[10] Malach R, Reppas JB, Benson RR, Kwong KK, Jiang H, Kennedy WA, Ledden PJ, Brady TJ, Rosen BR, and Tootell RB. (1995) Object-related activity revealed by functional magnetic resonance imaging in human occipital cortex. *Proc Natl Acad Sci U S A* 92:8135-8139.

[11] Chechik G, Globerson A, Anderson MJ, Young ED, Nelken I, and Tishby N. (2001) Groups redundancy measures reveal redundancy reduction along the auditory pathway. Advances in Neural Information Processing Systems 14. Cambridge, MA: MIT Press.
